# Rapidly Adapting Artificial Neural Networks for Autonomous Navigation

**Dean A. Pomerleau**
School of Computer Science
Carnegie Mellon University
Pittsburgh, PA 15213

## Abstract

The ALVINN (Autonomous Land Vehicle In a Neural Network) project addresses the problem of training artificial neural networks in real time to perform difficult perception tasks. ALVINN is a back-propagation network that uses inputs from a video camera and an imaging laser rangefinder to drive the CMU Navlab, a modified Chevy van. This paper describes training techniques which allow ALVINN to learn in under 5 minutes to autonomously control the Navlab by watching a human driver's response to new situations. Using these techniques, ALVINN has been trained to drive in a variety of circumstances including single-lane paved and unpaved roads, multilane lined and unlined roads, and obstacle-ridden on- and off-road environments, at speeds of up to 20 miles per hour.

## 1 INTRODUCTION

Previous trainable connectionist perception systems have often ignored important aspects of the form and content of available sensor data. Because of the assumed impracticality of training networks to perform realistic high level perception tasks, connectionist researchers have frequently restricted their task domains to either toy problems (e.g. the T-C identification problem [11] [6]) or fixed low level operations (e.g. edge detection [8]). While these restricted domains can provide valuable insight into connectionist architectures and implementation techniques, they frequently ignore the complexities associated with real world problems.

There are exceptions to this trend towards simplified tasks. Notable successes in high level domains such as speech recognition [12], character recognition [5] and face recognition [2] have been achieved using real sensor data. However, the results have come only in very controlled environments, after careful preprocessing of the input to segment and label the training exemplars. In addition, these successful connectionist perception systems have ignored the fact that sensor data normally becomes available gradually and not as a monolithic training set. In short, artificial neural networks previously have never been successfully trained

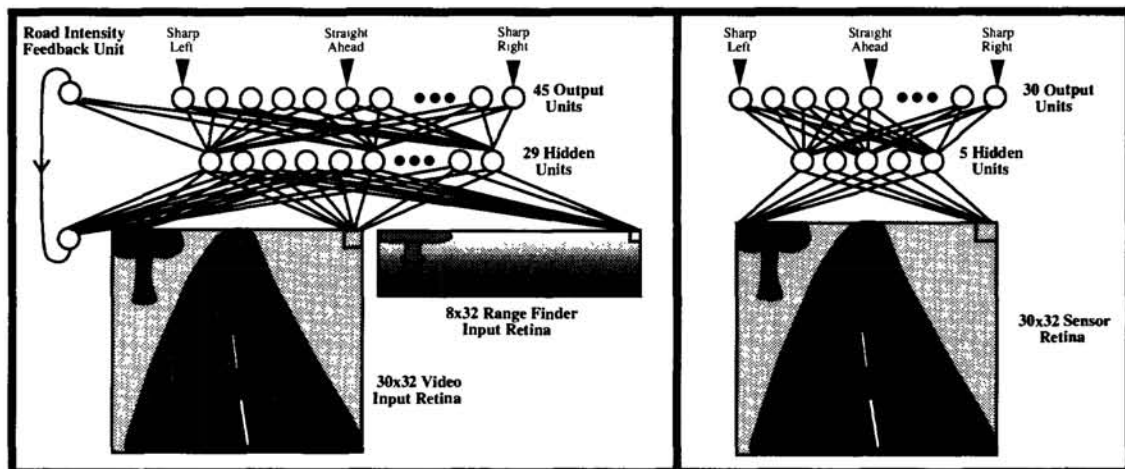

Figure 1: ALVINN's previous (left) and current (right) architectures

using sensor data in real time to perform a real world perception task.

The ALVINN (Autonomous Land Vehicle In a Neural Network) system remedies this short-coming. ALVINN is a back-propagation network designed to drive the CMU Navlab, a modified Chevy van. Using real time training techniques, the system quickly learns to au-tonomously control the Navlab by watching a human driver's reactions. ALVINN has been trained to drive in a variety of circumstances including single-lane paved and unpaved roads, multilane lined and unlined roads and obstacle ridden on- and off-road environments, at speeds of up to 20 miles per hour. This paper will primarily focus on improvements and extensions made to the ALVINN system since the presentation of this work at the 1988 NIPS conference [9].

## 2    NETWORK ARCHITECTURE

The current architecture for an individual ALVINN driving network is significantly simpler than the previous version (See Figure 1). The input layer now consists of a single 30x32 unit "retina" onto which a sensor image from either the video camera or the laser rangefinder is projected. Each of the 960 input units is fully connected to the hidden layer of 5 units, which is in turn fully connected to the output layer. The 30 unit output layer is a linear representation of the currently appropriate steering direction which may serve to keep the vehicle on the road or to prevent it from colliding with nearby obstacles[1]. The centermost output unit represents the "travel straight ahead" condition, while units to the left and right of center represent successively sharper left and right turns.

The reductions in network complexity over previous versions have been made in response to experience with ALVINN in actual driving situations. I have found that the distributed nature of the internal representation allows a network of only 5 hidden units to accurately drive in a variety of situations. I have also learned that multiple sensor inputs to a single network are redundant and can be eliminated. For instance, when training a network on a single-lane road, there is sufficient information in the video image alone for accurate driving. Similarly, for obstacle avoidance, the laser rangefinder image is sufficient and the video image

is superfluous. The road intensity feedback unit has been eliminated on similar grounds. In the previous architecture, it provided the network with the relative intensity of the road vs. the non-road in the previous image. This information was unnecessary for accurate road following, and undefined in new ALVINN domains such as off-road driving.

To drive the Navlab, an image from the appropriate sensor is reduced to 30 x 32 pixels and projected onto the input layer. After propagating activation through the network, the output layer's activation profile is translated into a vehicle steering command. The steering direction dictated by the network is taken to be the center of mass of the "hill" of activation surrounding the output unit with the highest activation level. Using the center of mass of activation instead of the most active output unit when determining the direction to steer permits finer steering corrections, thus improving ALVINN's driving accuracy.

## 3   TRAINING "ON-THE-FLY"

The most interesting recent improvement to ALVINN is the training technique. Originally, ALVINN was trained with backpropagation using 1200 simulated scenes portraying roads under a wide variety of weather and lighting conditions [9]. Once trained, the network was able to drive the Navlab at up to 1.8 meters per second (3.5 mph) along a 400 meter path through a wooded area of the CMU campus in weather which included snowy, rainy, sunny and cloudy situations.

Despite its apparent success, this training paradigm had serious shortcomings. It required approximately 6 hours of Sun-4 CPU time to generate the synthetic road scenes, and then an additional 45 minutes of Warp[2] computation time to train the network. Furthermore, while effective at training the network to drive on a single-lane road, extending the synthetic training paradigm to deal with more complex driving situations like multilane and off-road driving would have required prohibitively complex artificial scene generators.

I have developed a scheme called training "on-the-fly" to deal with these problems. Using this technique, the network learns to imitate a person as he drives. The network is trained with back-propagation using the latest video camera image as input and the person's current steering direction as the desired output.

There are two potential problems associated with this simple training on-the-fly scheme. First, since the person steers the vehicle down the center of the road during training, the network will never be presented with situations where it must recover from misalignment errors. When driving for itself, the network may occasionally stray from the road center, so it must be prepared to recover by steering the vehicle back to the middle of the road. The second problem is that naively training the network with only the current video image and steering direction may cause it to overlearn recent inputs. If the person drives the Navlab down a stretch of straight road near the end of training, the network will be presented with a long sequence of similar images. This sustained lack of diversity in the training set will cause the network to "forget" what it had learned about driving on curved roads and instead learn to always steer straight ahead.

Both problems associated with training on-the-fly stem from the fact that back-propagation requires training data which is representative of the full task to be learned. To provide the necessary variety of exemplars while still training on real data, the simple training on-the-

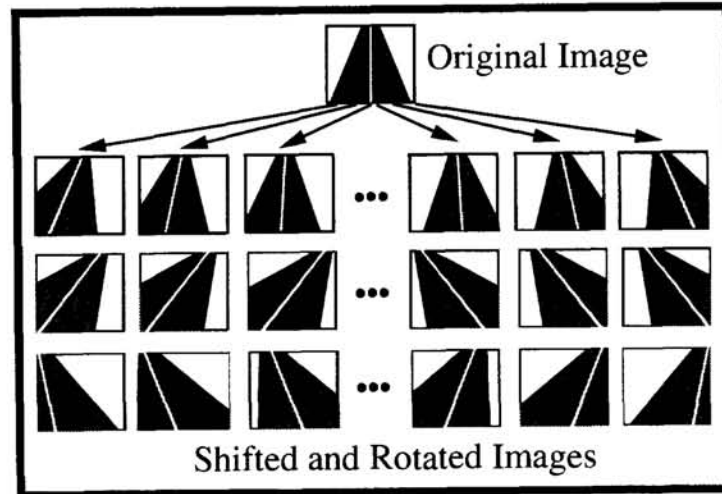

Figure 2: The single original video image is shifted and rotated to create multiple training exemplars in which the vehicle appears to be a different locations relative to the road.

fly scheme described above must be modified. Instead of presenting the network with only the current video image and steering direction, each original image is shifted and rotated in software to create 14 additional images in which the vehicle appears to be situated differently relative to the environment (See Figure 2). The sensor's position and orientation relative to the ground plane are known, so precise transformations can be achieved using perspective geometry. The correct steering direction as dictated by the driver for the original image is altered for each of the transformed images to account for the altered vehicle placement[3]. Using transformed training patterns allows the network to learn how to recover from driving errors. Also, overtraining on repetitive images is less of a problem, since the transformed training exemplars add variety to the training set. As additional insurance against the effects of repetitive exemplars, the training set diversity is further increased by maintaining a buffer of previously encountered training patterns.

In practice, training on-the-fly works as follows. A live sensor image is digitized and reduced to the low resolution image required by the network. This single original image is shifted and rotated 14 times to create 14 additional training exemplars[4]. Fifteen old exemplars from the current training set of 200 patterns are chosen and replaced by the 15 new exemplars. The 15 exemplars to be replaced in the training set are chosen on the basis of how closely they match the steering direction of one of the new tokens. Exchanging a new token for an old token with a similar steering direction helps maintain diversity in the training buffer during monotonous stretches of road by preventing novel older patterns from being replaced by recent redundant ones.

After this replacement process, one forward and one backward pass of the back-propagation algorithm is performed on the 200 exemplars to update the network's weights. The entire process is then repeated. The network requires approximately 50 iterations through this digitize-replace-train cycle to learn to drive in the domains that have been tested. Running

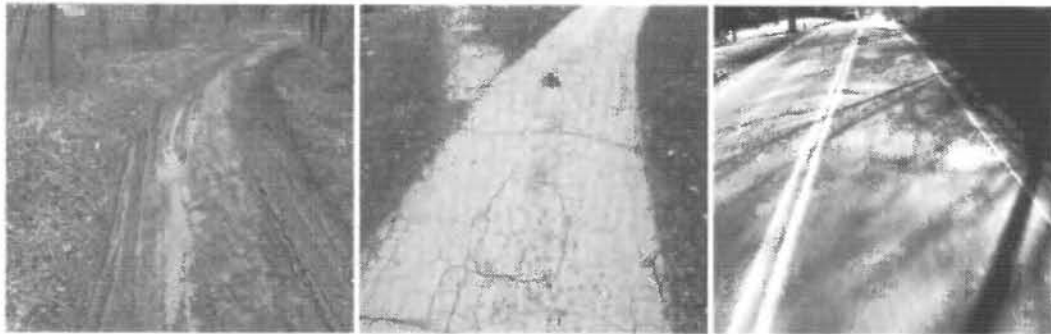

Figure 3: Video images taken on three of the test roads ALVINN has been trained to drive on. They are, from left to right, a single-lane dirt access road, a single-lane paved bicycle path, and a lined two-lane highway.

on a Sun-4, this takes about five minutes during which a person drives the Navlab at about 4 miles per hour over the training road.

## 4   RESULTS AND DISCUSSION

Once it has learned, the network can accurately traverse the length of road used for training and also generalize to drive along parts of the road it has never encountered under a variety of weather conditions. In addition, since determining the steering direction from the input image merely involves a forward sweep through the network, the system is able to process 25 images per second, allowing it to drive at up to the Navlab's maximum speed of 20 miles per hour[5]. This is over twice as fast as any other sensor-based autonomous system has driven the Navlab [3] [7].

The training on-the-fly scheme gives ALVINN a flexibility which is novel among autonomous navigation systems. It has allowed me to successfully train individual networks to drive in a variety of situations, including a single-lane dirt access road, a single-lane paved bicycle path, a two-lane suburban neighborhood street, and a lined two-lane highway (See Figure 3). Using other sensor modalities as input, including laser range images and laser reflectance images, individual ALVINN networks have been trained to follow roads in total darkness, to avoid collisions in obstacle rich environments, and to follow alongside railroad tracks. ALVINN networks have driven in each of these situations for up to 1/2 mile, until reaching a dead end or a difficult intersection. The development of a system for each of these domains using the "traditional approach" to autonomous navigation would require the programmer to 1) determine what features are important for the particular task, 2) program detectors (using statistical or symbolic techniques) for finding these important features and 3) develop an algorithm for determining which direction to steer from the location of the detected features.

In contrast, ALVINN is able to *learn* for each new domain what image features are important, how to detect them and how to use their position to steer the vehicle. Analysis of the hidden unit representations developed in different driving situations shows that the network forms detectors for the image features which correlate with the correct steering direction. When trained on multi-lane roads, the network develops hidden unit feature detectors for the lines painted on the road, while in single-lane driving situations, the detectors developed are

sensitive to road edges and road-shaped regions of similar intensity in the image. For a more detailed analysis of ALVINN's internal representations see [9] [10].

This ability to utilize arbitrary image features can be problematic. This was the case when ALVINN was trained to drive on a poorly defined dirt road with a distinct ditch on its right side. The network had no problem learning and then driving autonomously in one direction, but when driving the other way, the network was erratic, swerving from one side of the road to the other. After analyzing the network's hidden representation, the reason for its difficulty became clear. Because of the poor distinction between the road and the non-road, the network had developed only weak detectors for the road itself and instead relied heavily on the position of the ditch to determine the direction to steer. When tested in the opposite direction, the network was able to keep the vehicle on the road using its weak road detectors but was unstable because the ditch it had learned to look for on the right side was now on the left. Individual ALVINN networks have a tendency to rely on *any* image feature consistently correlated with the correct steering direction. Therefore, it is important to expose them to a wide enough variety of situations during training so as to minimize the effects of transient image features.

On the other hand, experience has shown that it is more efficient to train several domain specific networks for circumstances like one-lane vs. two-lane driving, instead training a single network for all situations. To prevent this network specificity from reducing ALVINN's generality, I am currently implementing connectionist and non-connectionist techniques for combining networks trained for different driving situations. Using a simple rule-based priority system similar to the subsumption architecture [1], I have recently combined a road following network and an obstacle avoidance network. The road following network uses video camera input to follow a single-lane road. The obstacle avoidance network uses laser rangefinder images as input. It is trained to swerve appropriately to prevent a collision when confronted with obstacles and to drive straight when the terrain ahead is free of obstructions. The arbitration rule gives priority to the road following network when determining the steering direction, except when the obstacle avoidance network outputs a sharp steering command. In this case, the urgency of avoiding an imminent collision takes precedence over road following and the steering direction is determined by the obstacle avoidance network. Together, the two networks and the arbitration rule comprise a system capable of staying on the road and swerving to prevent collisions.

To facilitate other rule-based arbitration techniques, I am currently adding to ALVINN a non-connectionist module which maintains the vehicle's position on a map. Knowing its map position will allow ALVINN to use arbitration rules such as "when on a stretch of two lane highway, rely primarily on the two lane highway network". This symbolic mapping module will also allow ALVINN to make high level, goal-oriented decisions such as which way to turn at intersections and when to stop at a predetermined destination.

Finally, I am experimenting with connectionist techniques, such as the task decomposition architecture [6] and the meta-pi architecture [4], for combining networks more seamlessly than is possible with symbolic rules. These connectionist arbitration techniques will enable ALVINN to combine outputs from networks trained to perform the same task using different sensor modalities and to decide when a new expert must be trained to handle the current situation.

### Acknowledgements

The principle support for the Navlab has come from DARPA, under contracts DACA76-85-C-0019, DACA76-85-C-0003 and DACA76-85-C-0002. This research was also funded in part by a grant from Fujitsu Corporation.

## Footnotes

[1]The task a particular driving network performs depends on the type of input sensor image and the driving situation it has been trained to handle.

[2]There was formerly a 100 MFLOP Warp systolic array supercomputer onboard the Navlab. It has been replaced by 3 Sun-4s, further necessitating the streamlined architecture described in the previous section.

[3]A simple steering model is used when transforming the driver's original direction. It assumes the "correct" steering direction is the one that will eliminate the additional vehicle translation and rotation introduced by the transformation and bringing the vehicle to the point the person was originally steering towards a fixed distance ahead of the vehicle.

[4]The shifts are chosen randomly from the range -1.25 to +1.25 meters and the rotations from the range -6.0 to +6.0 degrees.

[5]The Navlab has a hydraulic drive system which allows for very precise speed control, but which prevents the vehicle from driving over 20 miles per hour.

# References

[1] Brooks, R.A. (1986) A robust layered control system for a mobile robot. *IEEE Journal of Robotics and Automation*, vol. RA-2, no. 1, pp. 14-23, April 1986.

[2] Cottrell, G.W. (1990) Extracting features from faces using compression networks: Face, identity, emotion and gender recognition using holons. In Connectionist Models: Proc. of the 1990 Summer School, David Touretzky (Ed.), Morgan Kaufmann, San Mateo, CA.

[3] Crisman, J.D. and Thorpe C.E. (1990) Color vision for road following. In *Vision and Navigation: The CMU Navlab* Charles Thorpe (Ed.), Kluwer Academic Publishers, Boston, MA.

[4] Hampshire, J.B., Waibel A.H. (1989) The meta-pi network: Building distributed knowledge representations for robust pattern recognition. Carnegie Mellon Technical Report CMU-CS-89-166-R. August, 1989.

[5] LeCun, Y., Boser, B., Denker, J.S., Henderson, D., Howard, R.E., Hubbard, W., and Jackel, L.D. (1989) Backpropagation applied to handwritten zip code recognition. *Neural Computation 1(4)*.

[6] Jacobs, R.A., Jordan, M.I., Barto, A.G. (1990) Task decomposition through competition in a modular connectionist architecture: The what and where vision tasks. Univ. of Massachusetts Computer and Information Science Technical Report 90-27, March 1990.

[7] Kluge, K. and Thorpe C.E. (1990) Explicit models for robot road following. In *Vision and Navigation: The CMU Navlab* Charles Thorpe (Ed.), Kluwer Academic Publishers, Boston, MA.

[8] Koch, C., Bair, W., Harris, J.G., Horiuchi, T., Hsu, A. and Luo, J. (1990) Real-time computer vision and robotics using analog VLSI circuits. In *Advances in Neural Information Processing Systems, 2*, D.S. Touretzky (Ed.), Morgan Kaufmann, San Mateo, CA.

[9] Pomerleau, D.A. (1989) ALVINN: An Autonomous Land Vehicle In a Neural Network, *Advances in Neural Information Processing Systems, 1*, D.S. Touretzky (Ed.), Morgan Kaufmann, San Mateo, CA.

[10] Pomerleau, D.A. (1990) Neural network based autonomous navigation. In *Vision and Navigation: The CMU Navlab* Charles Thorpe (Ed.), Kluwer Academic Publishers, Boston, MA.

[11] Rumelhart, D.E., Hinton, G.E., and Williams, R.J. (1986) Learning internal representations by error propagation. In D.E. Rumelhart and J.L. McClelland (Eds.) *Parallel Distributed Processing: Explorations in the Microstructures of Cognition. Vol. 1: Foundations*. Bradford Books/MIT Press, Cambridge, MA.

[12] Waibel, A., Hanazawa, T., Hinton, G., Shikano, K., Lang, K. (1988) Phoneme recognition: Neural Networks vs. Hidden Markov Models. *Proceedings from Int. Conf. on Acoustics, Speech and Signal Processing*, New York, New York.